# Hybrid reinforcement learning and its application to biped robot control

**Satoshi Yamada, Akira Watanabe, Michio Nakashima**
{yamada, watanabe, naka}@bio.crl.melco.co.jp
Advanced Technology R&D Center
Mitsubishi Electric Corporation
Amagasaki, Hyogo 661-0001, Japan

## Abstract

A learning system composed of linear control modules, reinforcement learning modules and selection modules (a hybrid reinforcement learning system) is proposed for the fast learning of real-world control problems. The selection modules choose one appropriate control module dependent on the state. This hybrid learning system was applied to the control of a stilt-type biped robot. It learned the control on a sloped floor more quickly than the usual reinforcement learning because it did not need to learn the control on a flat floor, where the linear control module can control the robot. When it was trained by a 2-step learning (during the first learning step, the selection module was trained by a training procedure controlled only by the linear controller), it learned the control more quickly. The average number of trials (about 50) is so small that the learning system is applicable to real robot control.

## 1 Introduction

Reinforcement learning has the ability to solve general control problems because it learns behavior through trial-and-error interactions with a dynamic environment. It has been applied to many problems, *e.g.*, pole-balance [1], back-gammon [2], manipulator [3], and biped robot [4]. However, reinforcement learning has rarely been applied to real robot control because it requires too many trials to learn the control even for simple problems.

For the fast learning of real-world control problems, we propose a new learning system which is a combination of a known controller and reinforcement learning. It is called the hybrid reinforcement learning system. One example of a known controller is a linear controller obtained by linear approximation. The hybrid learning system

will learn the control more quickly than usual reinforcement learning because it does not need to learn the control in the state where the known controller can control the object.

A stilt-type biped walking robot was used to test the hybrid reinforcement learning system. A real robot walked stably on a flat floor when controlled by a linear controller [5]. Robot motions could be approximated by linear differential equations. In this study, we will describe hybrid reinforcement learning of the control of the biped robot model on a sloped floor, where the linear controller cannot control the robot.

## 2   Biped Robot

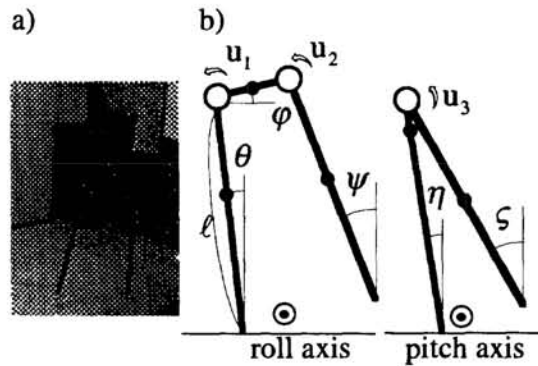

Figure 1: Stilt-type biped robot. **a)** a photograph of a real biped robot, **b)** a model structure of the biped robot. $u_1, u_2, u_3$ denote torques.

Figure 1-a shows a stilt-type biped robot [5]. It has no knee or ankle, has 1 m legs and weighs 33 kg. It is modeled by 3 rigid bodies as shown in Figure 1-b. By assuming that motions around a roll axis and those around a pitch axis are independent, 5-dimensional differential equations in a single supporting phase were obtained. Motions of the real biped robot were simulated by the combination of these equations and conditions at a leg exchange period. If angles are approximately zero, these equations can be approximated by linear equations. The following linear controller is obtained from the linear equations. The biped robot will walk if the angles of the free leg are controlled by a position-derivative (PD) controller whose desired angles are calculated as follows:

$$
\begin{aligned}
\bar{\varphi} &= \theta + \xi + \beta \\
\bar{\psi} &= \theta + 2\xi \\
\bar{\zeta} &= -A\dot{\eta} + \delta \\
A &= \sqrt{\frac{l}{g}},
\end{aligned}
\tag{1}
$$

where $\xi$, $\beta$, $\delta$, and $g$ are a desired angle between the body and the leg ($7°$), a constant to make up a loss caused by a leg exchange ($1.3°$), a constant corresponding to walking speed, and gravitational acceleration ($9.8$ ms$^{-2}$), respectively.

The linear controller controlled walking of the real biped robot on a flat floor [5]. However, it failed to control walking on a slope (Figure 2). In this study, the objective of the learning system was to control walking on the sloped floor shown in Figure 2-a.

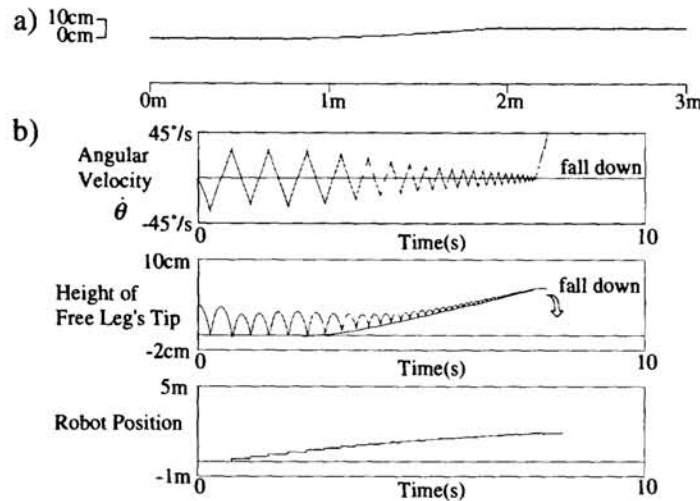

Figure 2: Biped robot motion on a sloped floor controlled by the linear controller. **a)** a shape of a floor, **b)** changes in angular velocity, height of free leg's tip, and robot position

## 3 Hybrid Reinforcement Learning

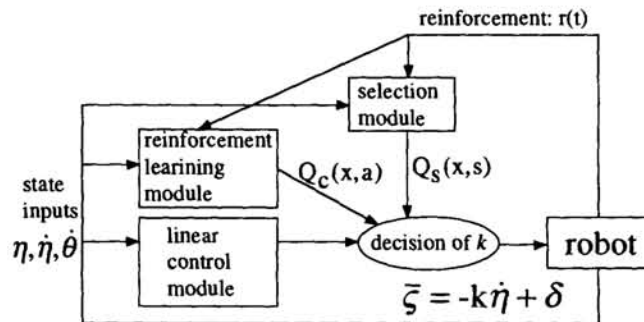

Figure 3: Hybrid reinforcement learning system.

We propose a hybrid reinforcement learning system to learn control quickly. The hybrid reinforcement learning system shown in Figure 3 is composed of a linear control module, a reinforcement learning module, and a selection module. The reinforcement learning module and the selection module select an action and a module dependent on their respective Q-values. This learning system is similar to the modular reinforcement learning system proposed by Tham [6] which was based on hierarchical mixtures of the experts (HME) [7]. In the hybrid learning system, the selection module is trained by Q-learning.

To combine the reinforcement learning with the linear controller described in (1), the output of the reinforcement learning module is set to $k$ in the adaptable equation for $\zeta$, $\zeta = -k\dot{\eta} + \delta$. The angle and the angular velocity of the supporting leg at the leg exchange period $(\eta, \dot{\eta}, \dot{\theta})$ are used as inputs. The $k$ values are kept constant until the next leg exchange. The reinforcement learning module is trained by "Q-sarsa" learning [8]. Q values are calculated by CMAC neural networks [9], [10].

The Q values for action $k$ ($Q_c(x,k)$) and those for module $s$ selection ($Q_s(x,s)$) are

calculated as follows:

$$Q_c(x,k) = \sum_{m,i} w_c(k,m,i,t)y(m,i,t)$$

$$Q_s(x,s) = \sum_{m,i} w_s(s,m,i,t)y(m,i,t), \qquad (2)$$

where $w_c(k,m,i,t)$ and $w_s(s,m,i,t)$ denote synaptic strengths and $y(m,i,t)$ represents neurons' outputs in CMAC networks at time $t$.

Modules were selected and actions performed according to the $\varepsilon$-greedy policy [8] with $\varepsilon = 0$.

The temporal difference (TD) error for the reinforcement learning module $(\hat{r}_c(t))$ is calculated by

$$\hat{r}_c(t) = \begin{cases} 0 & sel(t) = lin \\ r(t) + Q_c(x(t+1), per(t+1)) - Q_c(x(t), per(t)) & \begin{aligned} sel(t) &= rein \\ sel(t+1) &= rein \end{aligned} \\ r(t) + Q_s(x(t+1), sel(t+1)) - Q_c(x(t), per(t)), & \begin{aligned} sel(t) &= rein \\ sel(t+1) &= lin \end{aligned} \end{cases}$$

$$(3)$$

where $r(t)$, $per(t)$, $sel(t)$, $lin$ and $rein$ denote reinforcement signals ($r(t) = -1$ if the robot falls down, 0 otherwise), performed actions, selected modules, the linear control module and the reinforcement learning module, respectively.

TD error $(\hat{r}_t(t))$ calculated by $Q_s(x,s)$ is considered to be a sum of TD error caused by the reinforcement learning module and that by the selection module. TD error $(\hat{r}_s(t))$ used in the selection-module's learning is calculated as follows:

$$\begin{aligned} \hat{r}_s(t) &= \hat{r}_t(t) - \hat{r}_c(t) \\ &= r(t) + \gamma Q_s(x(t+1), sel(t+1)) - Q_s(x(t), sel(t)) - \hat{r}_c(t), \qquad (4) \end{aligned}$$

where $\gamma$ denotes a discount factor.

The reinforcement learning module used replacing eligibility traces $(e_c(k,m,i,t))$ [11]. Synaptic strengths are updated as follows:

$$w_c(k,m,i,t+1) = w_c(k,m,i,t) + \alpha_c \hat{r}_c(t)e_c(k,m,i,t)/n_t$$

$$w_s(s,m,i,t+1) = \begin{cases} w_s(s,m,i,t) + \alpha_s \hat{r}_s(t)y(m,i,t)/n_t & s = sel(t) \\ w_s(s,m,i,t) & \text{otherwise} \end{cases}$$

$$e_c(k,m,i,t) = \begin{cases} 1 & k = per(t), y(m,i,t) = 1 \\ 0 & k \neq per(t), y(m,i,t) = 1 \\ \lambda e_c(k,m,i,t-1) & \text{otherwise} \end{cases} \qquad (5)$$

where $\alpha_c$, $\alpha_s$, $\lambda$ and $n_t$ are a learning constant for the reinforcement learning module, that for the selection module, decay rates and the number of tilings, respectively.

In this study, the CMAC used 10 tilings. Each of the three dimensions was divided into 12 intervals. The reinforcement learning module had 5 actions ($k = 0, A/2, A, 3A/2, 2A$). The parameter values were $\alpha_s = 0.2$, $\alpha_c = 0.4$, $\lambda = 0.3$, $\gamma = 0.9$ and $\delta = 0.05$. Each run consisted of a sequence of trials, where each trial began with robot state of position=0, $-5° < \theta < -2.5°, 1.5° < \eta < 3°, \varphi = \theta + \xi, \psi = \varphi + \xi, \zeta = \eta + 2°, \dot{\theta} = \dot{\varphi} = \dot{\psi} = \dot{\eta} = \dot{\zeta} = 0$, and ended with a failure signals indicating robot's falling down. Runs were terminated if the number of walking steps of three consecutive trials exceeded 100. All results reported are an average of 50 runs.

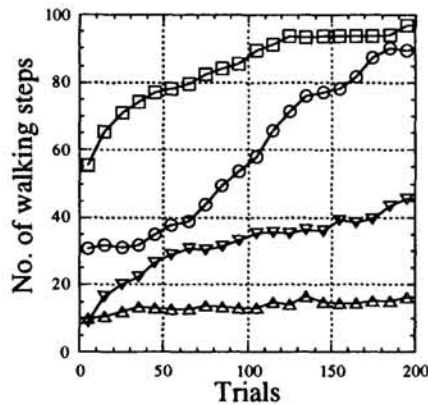

Figure 4: Learning profiles for control of walking on the sloped floor. (○) hybrid reinforcement learning, (□) 2-step hybrid reinforcement learning, (▽) reinforcement learning and (△) HME-type modular reinforcement learning

## 4    Results

Walking control on the sloped floor (Figure 2-a) was first trained by the usual reinforcement learning. The usual reinforcement learning system needed many trials for successful termination (about 800, see Figure 4(▽)). Because the usual reinforcement learning system must learn the control for each input, it requires many trials.

Figure 4(○) also shows the learning curve for the hybrid reinforcement learning. The hybrid system learned the control more quickly than the usual reinforcement learning (about 190 trials). Because it has a higher probability of succeeding on the flat floor, it learned the control quickly. On the other hand, HME-type modular reinforcement learning [6] required many trials to learn the control (Figure 4(△)).

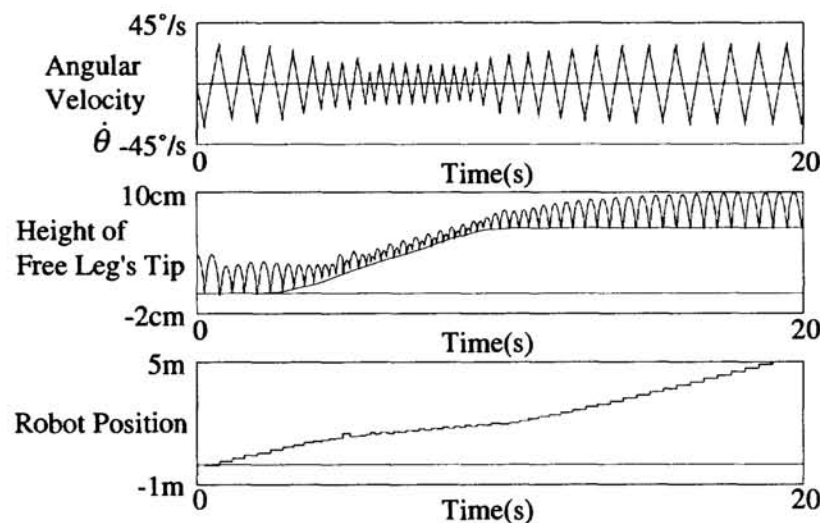

Figure 5: Biped robot motion controlled by the network trained by the 2-step hybrid reinforcement learning.

In order to improve the learning rate, a 2-step learning was examined. The 2-step learning is proposed to separate the selection-module learning from the reinforcement-learning-module learning. In the 2-step hybrid reinforcement learning, the selection module was first trained by a special training procedure in which the robot was controlled only by the linear control module. And then the network was trained by the hybrid reinforcement learning. The 2-step hybrid reinforcement learning learned the control more quickly than the 1-step hybrid reinforcement learning (Figure 4(□)). The average number of trials were about 50. The hybrid learning system may be applicable to the real biped robot.

Figure 5 shows the biped robot motion controlled by the trained network. On the slope, the free leg's lifting was magnified irregularly (see changes in the height of the free leg's tip of Figure 5) in order to prevent the reduction of an amplitude of walking rhythm. On the upper flat floor, the robot was again controlled stably by the linear control module.

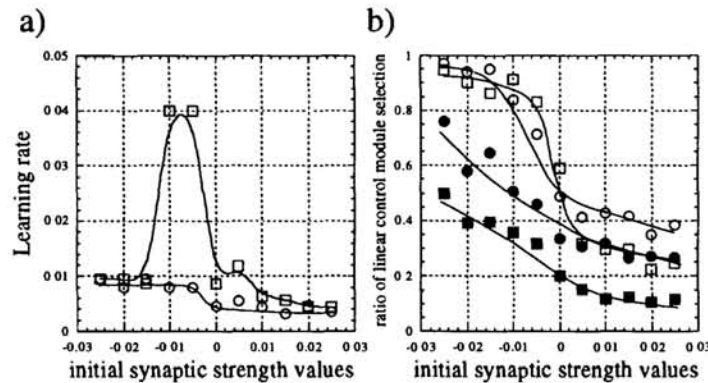

Figure 6: Dependence of (a) the learning rate and (b) the selection ratio of the linear control module on the initial synaptic strength values ($w_s(rein, m, i, 0)$). **(a)** learning rate of (○) the hybrid reinforcement learning, and (□) the 2-step hybrid reinforcement learning. The learning rate is defined as the inverse of the number of trials where the average walking steps exceed 70. **(b)** the ratio of the linear-control-module selection. Circles represent the selection ratio of the linear control module when controlled by the network trained by the hybrid reinforcement learning, rectangles represent that by the 2-step hybrid reinforcement learning. Open symbols represent the selection ratio on the flat floor, closed symbols represent that on the slope.

The dependence of learning characteristics on initial synaptic strengths for the reinforcement-learning-module selection ($w_s(rein, m, i, 0)$) was considered (other initial synaptic strengths were 0). If initial values of $w_s(rein, m, i, t)$ ($w_s(rein, m, i, 0)$) are negative, the Q-values for the reinforcement-learning-module selection ($Q_s(x, rein)$) are smaller than $Q_s(x, lin)$ and then the linear control module is selected for all states at the beginning of the learning. In the case of the 2-step learning, if $w_s(rein, m, i, 0)$ are given appropriate negative values, the reinforcement learning module is selected only around failure states, where $Q_s(x, lin)$ is trained in the first learning step, and the linear control module is selected otherwise at the beginning of the second learning step. Because the reinforcement learning module only requires training around failure states in the above condition, the 2-

step hybrid system is expected to learn the control quickly. Figure 6-a shows the dependence of the learning rate on the initial synaptic strength values. The 2-step hybrid reinforcement learning had a higher learning rate when $w_s(rein, m, i, 0)$ were appropriate negative values (-0.01 ~ -0.005). The trained system selected the linear control module on the flat floor (more than 80%), and selected both modules on the slope (see Figure 6-b), when $w_s(rein, m, i, 0)$ were negative.

Three trials were required in the first learning step of the 2-step hybrid reinforcement learning. In order to learn the Q-value function around failure states, the learning system requires 3 trials.

## 5  Conclusion

We proposed the hybrid reinforcement learning which learned the biped robot control quickly. The number of trials for successful termination in the 2-step hybrid reinforcement learning was so small that the hybrid system is applicable to the real biped robot. Although the control of real biped robot was not learned in this study, it is expected to be learned quickly by the 2-step hybrid reinforcement learning. The learning system for real robot control will be easily constructed and should be trained quickly by the hybrid reinforcement learning system.

## References

[1] Barto, A. G., Sutton, R. S. and Anderson, C. W.: Neuron like adaptive elements that can solve difficult learning control problems, *IEEE Trans. Sys. Man Cybern.*, Vol. SMC-13, pp. 834–846 (1983).

[2] Tesauro, G.: TD-gammon, a self-teaching backgammon program, achieves master-level play, *Neural Computation*, Vol. 6, pp. 215–219 (1994).

[3] Gullapalli, V., Franklin, J. A. and Benbrahim, H.: Acquiring robot skills via reinforcement learning, *IEEE Control System*, Vol. 14, No. 1, pp. 13–24 (1994).

[4] Miller, W. T.: Real-time neural network control of a biped walking robot, *IEEE Control Systems*, Vol. 14, pp. 41–48 (1994).

[5] Watanabe, A., Inoue, M. and Yamada, S.: Development of a stilts type biped robot stabilized by inertial sensors (in Japanese), in *Proceedings of 14th Annual Conference of RSJ*, pp. 195–196 (1996).

[6] Tham, C. K.: Reinforcement learning of multiple tasks using a hierarchical CMAC architecture, *Robotics and Autonomous Systems*, Vol. 15, pp. 247–274 (1995).

[7] Jordan, M. I. and Jacobs, R. A.: Hierarchical mixtures of experts and the EM algorithm, *Neural Computation*, Vol. 6, pp. 181–214 (1994).

[8] Sutton, R. S.: Generalization in reinforcement learning: successful examples using sparse coarse coding, *Advances in NIPS*, Vol. 8, pp. 1038–1044 (1996).

[9] Albus, J. S.: A new approach to manipulator control: The cerebellar model articulation controller (CMAC), *Transaction on ASME J. Dynamical Systems, Measurement, and Controls*, pp. 220–227 (1975).

[10] Albus, J. S.: Data storage in the cerebellar articulation controller (CMAC), *Transaction on ASME J. Dynamical Systems, Measurement, and Controls*, pp. 228–233 (1975).

[11] Singh, S. P. and Sutton, R. S.: Reinforcement learning with replacing eligibility traces, *Machine Learning*, Vol. 22, pp. 123–158 (1996).
